# Performance of a Stochastic Learning Microchip

Joshua Alspector, Bhusan Gupta,* and Robert B. Allen
Bellcore, Morristown, NJ 07960

We have fabricated a test chip in 2 micron CMOS that can perform supervised learning in a manner similar to the Boltzmann machine. Patterns can be presented to it at 100,000 per second. The chip learns to solve the XOR problem in a few milliseconds. We also have demonstrated the capability to do unsupervised competitive learning with it. The functions of the chip components are examined and the performance is assessed.

## 1. INTRODUCTION

In previous work,[1] [2] we have pointed out the importance of a local learning rule, feedback connections, and stochastic elements[3] for making learning models that are electronically implementable. We have fabricated a test chip in 2 micron CMOS technology that embodies these ideas and we report our evaluation of the microchip and our plans for improvements.

Knowledge is encoded in the test chip by presenting digital patterns to it that are examples of a desired input-output Boolean mapping. This knowledge is learned and stored entirely on chip in a digitally controlled synapse-like element in the form of connection strengths between neuron-like elements. The only portion of this learning system which is off chip is the VLSI test equipment used to present the patterns.

This learning system uses a modified Boltzmann machine algorithm[3] which, if simulated on a serial digital computer, takes enormous amounts of computer time. Our physical implementation is about 100,000 times faster. The test chip, if expanded to a board-level system of thousands of neurons, would be an appropriate architecture for solving artificial intelligence problems whose solutions are hard to specify using a conventional rule-based approach. Examples include speech and pattern recognition and encoding some types of expert knowledge.

## 2. CHIP COMPONENTS

Fig. 1 is a photograph of the silicon chip. It contains various test structures, the largest of which, in the lower left, is a neural-style learning network composed of 6 neurons, each with its own noise amplifier, and 15 bidirectional synapses which potentially allow the network to be fully connected. In order to study these components separately, there is a also a noise amplifier in the upper left corner of the chip, a neuron in the upper right, and 2 synapses in the lower right.

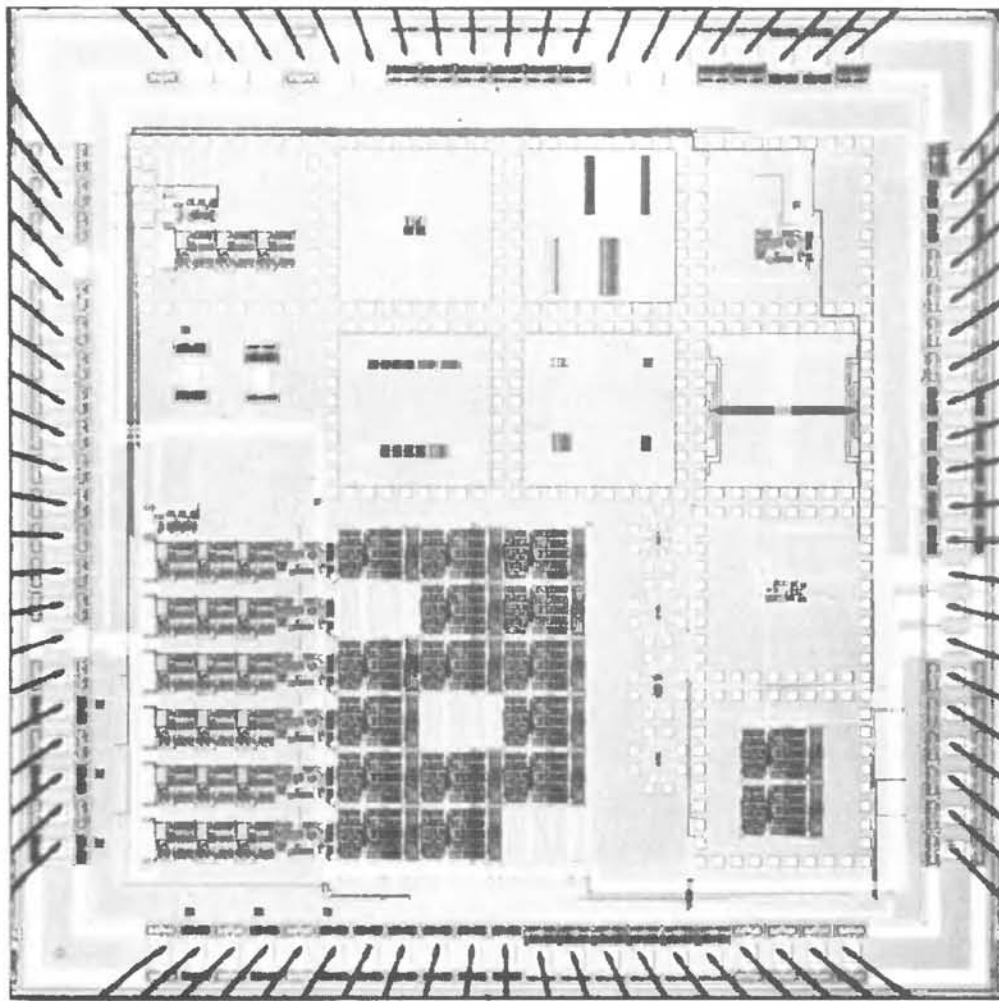

Figure 1. Photograph of Test Chip Containing a Learning Network in Lower Left.

## 2.1 Neuron

The electronic neuron performs the physical computation:

$$activation = f\,(\Sigma w_{ij}\,s_j + noise\,) = f\,(gain * net_i\,)$$

where $f$ is a monotonic non-linear function such as *tanh*. In some of our computer simulations this is a step function corresponding to a high value of *gain*. The signal from other neurons to neuron $i$ is the sum of neural states $s_i$ giving input weighted by the connection strengths $w_{ij}$, while the *noise* simulates a temperature in a physical thermodynamic system. Their sum is the effective net input $net_i$.

The model neuron is a double differential amplifier as shown in Fig. 2. Noise and signal have separate differential inputs and are summed at low gain. The differential outputs of this summing stage are converted to a single output by a high gain stage before being fed into a switching arrangement. This selects either the net input or an external clamping signal which forces the neuron into a desired state. The output of the switch is then

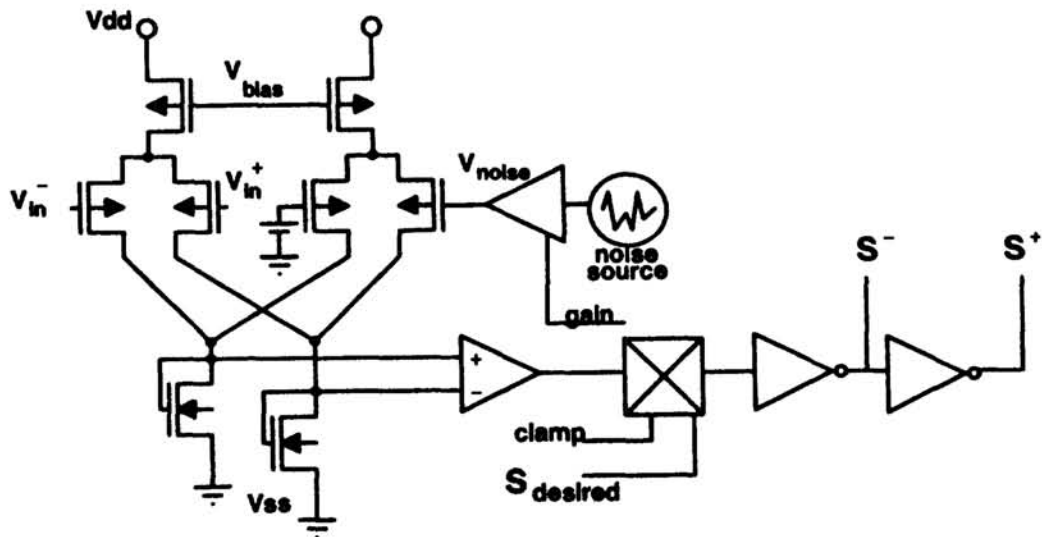

Figure 2. Circuitry of Electronic Analog Neuron.

further amplified before driving the network. The final output approximates a two-state binary neuron.

**2.2 Noise amplifier**

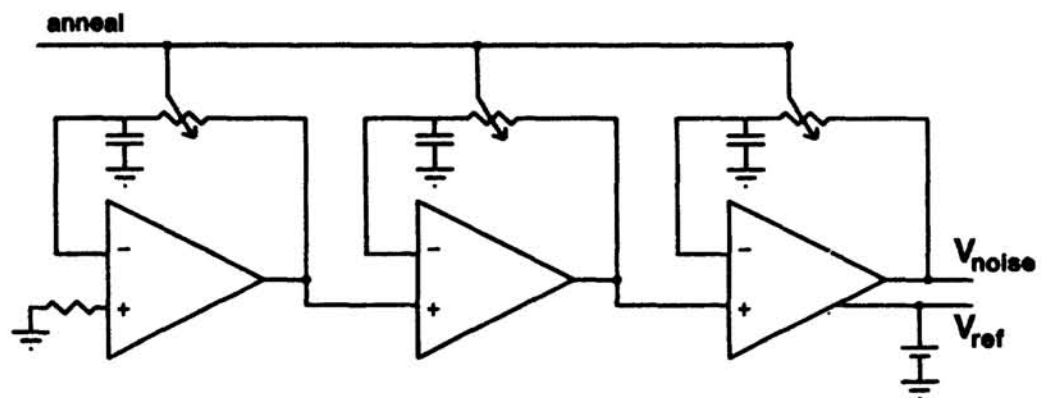

Figure 3. Block Diagram of Noise Amplifier.

Fig. 3 is a block diagram of the noise amplifier. The original idea was to amplify the thermal noise in the channel of a transistor with a gain of nearly a million but to stabilize the dc output using low pass negative feedback in 3 stages. By controlling the feedback, one could control both the bandpass of the noise signal as well as the gain to provide for annealing the temperature (amount of noise) as required by the Boltzmann machine algorithm.[3] Unfortunately this amplifier proved unstable at high gain values leading to oscillations of a few MHz which were highly correlated among all the noise amplifiers in the network. In spite of this undesirable correlation in the noise signals, the network was still able to learn (see section 3). Rather than a slow "annealing", we used a rapid "heating" and "flash freezing" of the network to randomize ·it. This was done by momentarily opening a "noise on" switch during the time allotted for annealing. Learning was also demonstrated by clamping the free running neurons momentarily to a pseudo-random state and then releasing them to allow the network to settle.

### 2.3 Synapse

Fig. 4 is a block diagram of the digitally controlled electronic synapse. The weights are stored as a sign and four bits of magnitude in five flip-flops arranged as an up-down counter. The correlation logic tests whether the two neurons that the synapse connects have the same binary state (correlated) or not at the end of the anneal cycle. If the neurons are correlated in the "teacher" phase (when the teacher is clamping the output neurons in the correct state) and not in the "student" phase (when the output neurons are running free), then a signal to the counter increments the weight by one. If the reverse is true, the counter is decremented. If the "teacher" and "student" phase have the same correlation, no change is made.

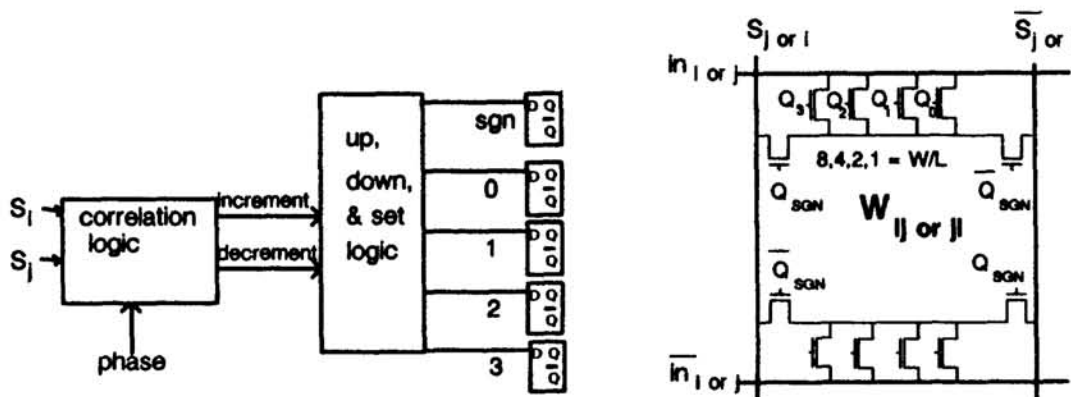

Figure 4. Block Diagram of Synapse.

The digital weight is converted to an analog conductance by a set of pass transistors with graduated binary conductance ratios. Measurements confirmed that the synapse conductance increased monotonically from a value of -15 though +15 as the counter was incremented. The -0 value, when loaded into the synapse, disconnected that link. We usually initialized all the weights to +0 before learning.

### 3. PERFORMANCE EVALUATION OF NETWORK

#### 3.1 XOR tests

The most difficult test for our 6 neuron network was to have it learn the exclusive-OR function. The network was arranged with 2 input neurons, 2 hidden neurons, and 1 output neuron as shown in Fig. 5. There is also a so-called 'true' neuron which is always clamped on. The negative of the weights from that neuron provide the threshold for the other neurons. The exclusive-OR function is of historical interest because the neural models of the 1960's could not learn it.[4] [5] This is because those learning algorithms did not work when there was a layer of hidden neurons. Networks with only a single layer of modifiable weights could learn the logical OR function but not the exclusive-OR (XOR). The truth table in Fig. 5 shows that the XOR is 1 (on or true) when either one of the two inputs is 1, but not when both are 1. However, recent algorithms such as the Boltzmann machine are able to learn with a hidden layer and hence can solve the XOR.

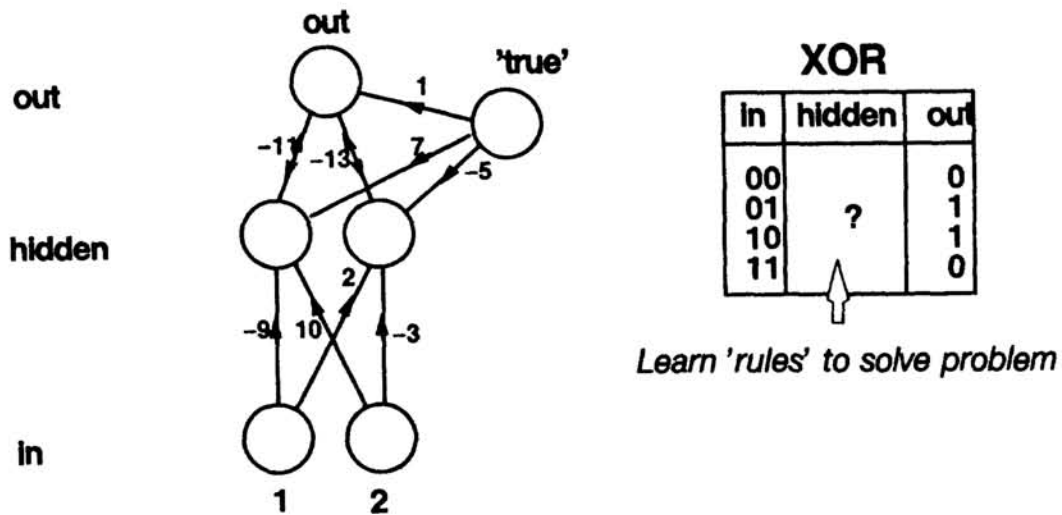

Figure 5. 2-2-1 Network to Learn XOR.

To teach a network to be an XOR, we start with a blank slate where all the weights are zero and then present the patterns of 1's and 0's in the figure with the teacher alternately clamping the output to the correct state and letting it run free. On each presentation, the network is jittered by noise and correlations are counted by each synapse. At the end of each teacher-student cycle, weights are adjusted.

Tests of the chip were conducted using an HP 8180A data generator to present digital patterns to the chip, an HP 8182 data analyzer to capture the chip's digital outputs, and an HP 54112A digitizing oscilloscope to capture waveforms. Analog waveforms were generated using an HP 8770A arbitrary waveform synthesizer feeding a Comlinear E201I amplifier. These instruments were controlled by an HP 9836 computer running UNIX with test programs written in C.

A pattern presentation phase consisted of five subphases and hence five clock cycles of the data generator. The input and/or output pattern to be presented to the clamped neurons is present during all five cycles. The first cycle presents noise or an annealing waveform to the network. The second cycle sends a signal to each synapse to count correlations. The fourth cycle can be used to send a signal to each synapse to adjust weights. This is usually done only after two 5 cycle phases, one for the "teacher" phase and one for the "student" phase. Thus, during learning, ten digital words were used in the data generator for each pattern presentation.

In addition to presenting patterns, digital weights can also be read into the chip with a similar 5 cycle phase. This uses the flip-flop storage arranged as a shift register for weight storage and readout. Because the memory of the data generator was only 1024 bits deep, we would present only 66 patterns (660 words) each time the data generator was loaded by the control computer. The remaining memory was used to initialize the network to its previous value after the destructive readout of weights. In this way, performance of the network was monitored after sets of 66 pseudo-randomly selected patterns. 100 test patterns could also be presented, without learning, to see what performance the network achieved at that point.

For the XOR, we organized the connectivity as in Fig. 5. For example, the connections between input and output neurons were fixed at zero. In order to test the settling of the network, we loaded a set of synapse weights that were learned in one of the computer simulations. We then checked the settling times of the network for various transitions of input states. These varied from 130 to 1700 nanoseconds, with most transitions in the 250 to 600 nanosecond range. The shortest time is a simple settling of the neuron amplifier while the longest time represents several loops of settling of the network before a stable state is found.

For the learning trials, we initialized all weights to zero. Fig. 6 shows three learning curves for a 2-2-1 XOR network (Fig. 5). At first the network performs at chance but it soon learns all the patterns. The values of the weights (which have an accuracy of 4 bits plus a sign) after learning are also shown for one of the trials.

The chip had an easier time learning the XOR function in a network with only one hidden unit provided there were also direct connections from input to output as shown in the inset of Fig. 7. This also demonstrates the flexibility of the connectivity on the chip which would not be possible if we organized it as a strictly layered network. The figure shows the learning curves at various speeds of pattern presentation from 500 to 256,000 patterns per second. The clock rate of the data generator at the highest speed was 2.56 MHz so that the time during which noise was applied was only 400 nanoseconds. The noise amplifier often did not produce an excursion of neural states at these frequencies

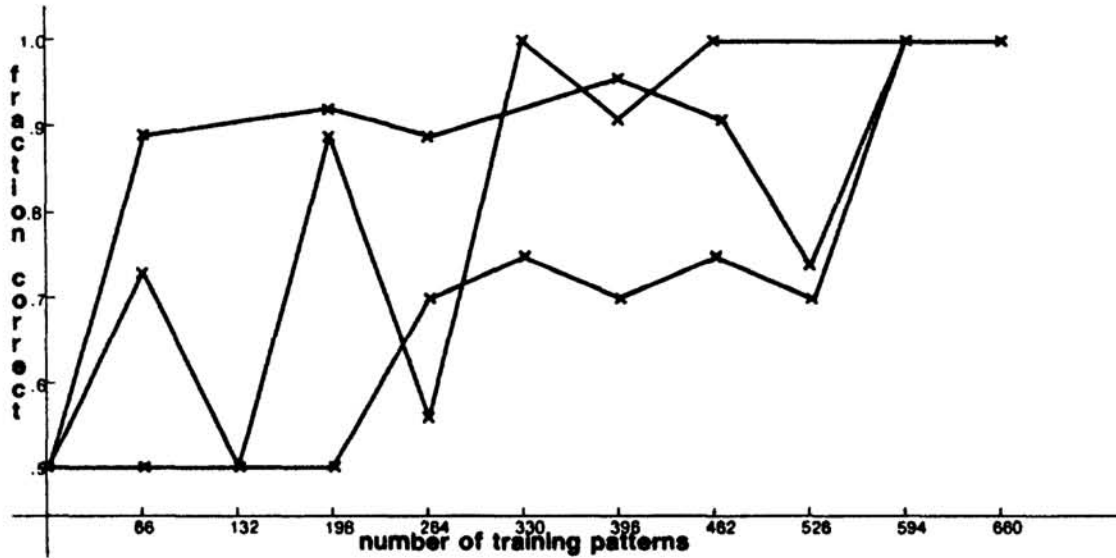

Figure 6.  Proportion Correct for On-chip Learning vs. Patterns Presented.

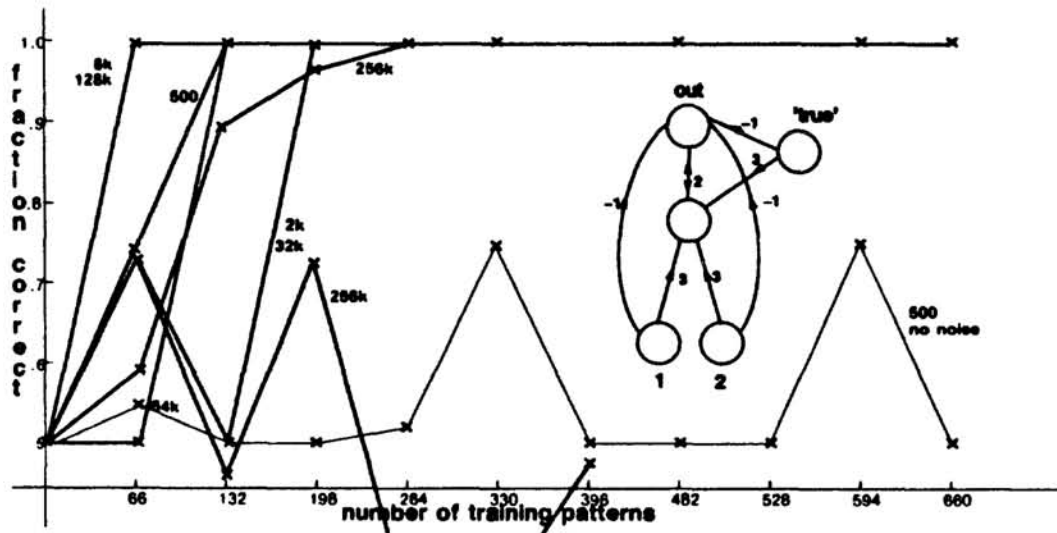

Figure 7.  Learning Curves for 2-1-1 XOR at Various Speeds.

effectively limiting learning above this rate.  We could have increased the rate by compressing the five cycle phase to three or by random clamping of free running neurons, but probably not by an order of magnitude.  Note that noise is necessary for learning by this system as shown by the curve at 500 Hz without noise.

Fig. 8 is an oscilloscope trace of the 4 neural states as a function of time during the pattern presentations.

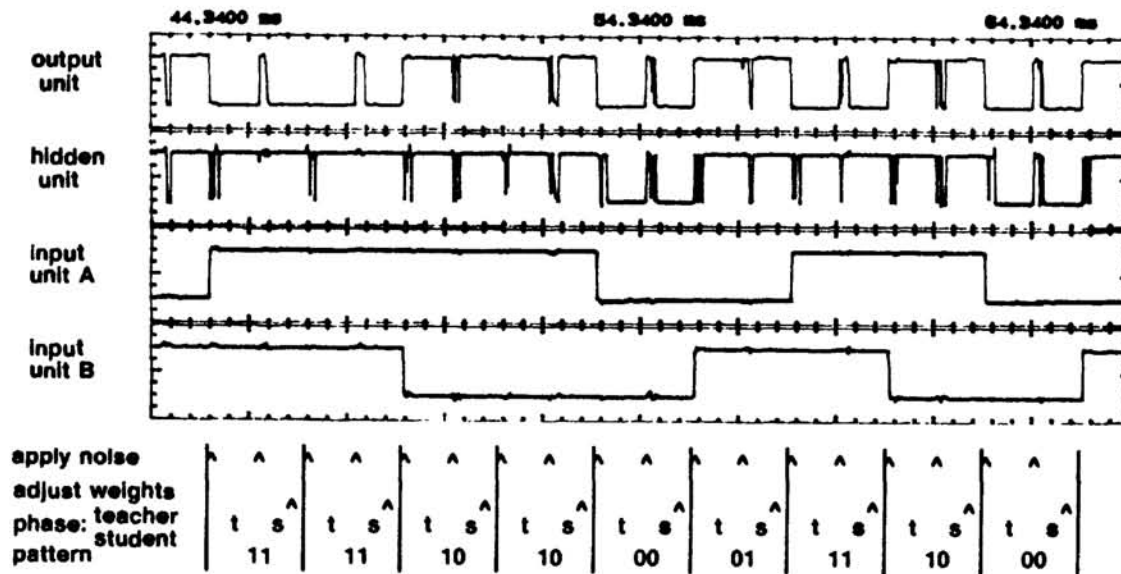

Figure 8. Neural States during Learning.

The time during which noise is applied is apparent from the rapid changes of state in the hidden neuron and also in the output neuron when it is not clamped. Since each pattern presentation can take as little as 5 microseconds, the XOR function can be learned in a few milliseconds. A pattern presentation on a 1 MIP serial computer such as a VAX 11/780 takes about 0.5 seconds with our simulation software.

## 3.2 Unsupervised Learning

So far, we have described only supervised learning procedures, but the chip can also do unsupervised learning which has no teacher. Nevertheless, the network can learn to classify input patterns according to their similarity to one another. We set the chip connectivity as in Fig. 9 with 4 input neurons and 2 output neurons arranged so that they strongly inhibit each other to form a 'competitive' layer. With noise, this output layer performs a 'winner-take-all' function in that the output neuron which has the strongest net input is on and the other is off. This is because they inhibit each other strongly (are connected to each other with a large negative weight) so that only one can be on. The usual supervised learning rule was effectively simplified by removing the teacher requirement so that correlations always increment weights. Specifically, we stored a comparison pattern in the student phase which consisted of the 'on' state for the two competitive neurons and 'off' for all the input neurons. We then presented patterns to the chip with the "teacher" phase signal on. This has the effect of always decrementing the competitive connections which therefore remain at the lower limit of -15 since it is not possible to have more correlations than the stored "student" phase correlation. On the other hand, the stored "student" phase correlation for the weights leading from the input

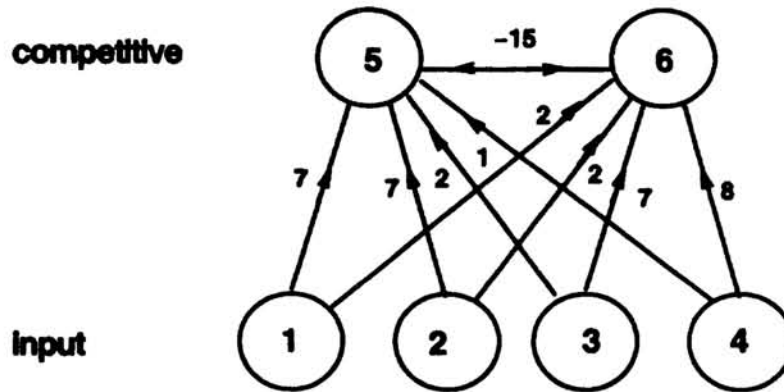

Figure 9.  A Competitive Learning Network.

to the competitive layer is zero.  Then, the winning output neuron will always be correlated with those input neurons which are 'on' and hence these weights will be incremented.  A decay signal decremented weights occasionally to keep them from growing too large.  The net effect of such a procedure is for the output neurons to classify the input space among themselves, such that each responds to a particular neighborhood of similar patterns.[2]

To demonstrate competitive learning, an input set was prepared such that the four input bits were not quite random.  We picked two input neurons to represent 'left' and the other two to represent 'right'.  Patterns were never used with an equal number of left and right neurons on.  Eventually one of the two output neurons responded to left weighted patterns and the other to right weighted patterns.  Fig. 9 shows one set of weights which were obtained.  Therefore the chip learned left from right although nothing in its wiring predisposed it in any way.

### 3.3  Computer Simulations of Chip Test Conditions

Computer tests were conducted which simulated limitations of the operating chip such as correlated noise.  Table 1 presents summaries of 10 replications of 2000 pattern presentations across 5 testing conditions.  The Table reports the mean percent correct on the last 100 patterns and, in parentheses, the number of networks which reached 100% performance during at least one block of 100 pattern presentations.  The first line of the table shows the performance of the network with no noise.  In the next four lines, two parameters of the noise were varied yielding 4 conditions.  Specifically, noise was either correlated or uncorrelated across neurons and it was either presented as a single pulse in a "flash freeze" schedule or following a broad annealing schedule.

The 2-1-1 XOR, in which the inputs are directly connected to the outputs, demonstrated very good performance across conditions. Indeed, additional tests of the 2-1-1 in the no-noise condition showed that within 10k patterns all networks reached 100%. This suggests there are deterministic solutions for the 2-1-1.

TABLE 1. Results of Computer Simulations.

| noise | schedule | 2-1-1 XOR | 2-2-1 XOR | 4-4-1 parity |
|-------|----------|-----------|-----------|--------------|
| no noise | - | 92(9) | 67(0) | 72(0) |
| correlated | flash freeze | 95(9) | 83(5) | 71(0) |
| correlated | anneal temperature | 99(10) | 78(2) | 74(0) |
| uncorrelated | flash freeze | 99(10) | 84(4) | 67(0) |
| uncorrelated | anneal temperature | 99(10) | 85(5) | 79(0) |
| no noise | anneal gain | 99(9) | 81(4) | 85(2) |

The 2-2-1 networks learned to only 67% correct without noise. Learning with correlated noise degraded performance compared to learning with uncorrelated noise. While the chip contained only 6 neurons it was of interest to consider how limitations such as those studied here might affect solutions to larger problems. Thus, the solution to parity problems were considered and are included in the table.

It is worth noting that the full complexity of the chip's settling and noise distribution is not captured in the discrete time simulations on the computer. The fact that we do not use a circuit simulation may account for some of the differences between the simulations and chip performance. It is interesting to note that learning by the chip was generally faster than learning by the simulation program and that the chip seemed to require noise for learning more than the simulator.

We also considered a system without random noise in which we annealed the inverse gain of the neurons like a temperature through a broad annealing schedule covering the values previously examined[2]. As shown in the last line of the Table this performed comparably to temperature annealing reported above. 10 runs of a 2-2-1 XOR gave a mean performance of 81% with 4 networks reaching 100%. On the 4-4-1 parity problem the mean performance was better than the results of annealing temperature. The mean performance was 85% and 2 networks reached 100%. For still larger problems, such as 6-8-1 parity, performance was comparable to annealing with noise.

## 4. FUTURE DIRECTIONS

### 4.1 Applications of Learning Systems

Learning systems give us a way to encode knowledge as a set of training examples rather than as a set of rules. Learned behavior emerges from the training set in ways that depend on the input representation, the network architecture, and the learning procedure. This technique is suitable for problem domains where there are too many rules or where the rules are not known. Two general categories of problems suitable for learning

systems are pattern recognition and some types of expert systems.

Pattern recognition of something like an oak leaf is difficult because of the many variations a rule-based system would have to consider even when variations of scale, rotation, and translation are accounted for. Yet, it is quite easy to give a learning system many training examples of oak leaves. Scale, rotation, and translation invariance can be built into the network structure. Similarly, recognition of speech sounds is difficult, but many training examples exist. Here also, pre-processing of the auditory data is important to obtain a useful representation. Another pattern learning task useful in telecommunications is learning the codebook for vector quantization in a real-time visual data compression system.[6]

Expert knowledge is often easier to encode by training examples as well. Experts often do not know the rules they use to troubleshoot equipment or give advice. Again, it is quite easy, by taking a history of such advice, to build a large database of training examples. As knowledge changes, training is a more graceful way of updating a knowledge base than changing the rules. In telephone networks, fault handling or traffic routing are examples of problems for which training is a suitable way of encoding knowledge.

### 4.2 Future Large-Scale Learning Systems

Because training takes too much computer time in a simulation, physical implementations of learning systems such as ours are necessary for speed. It takes several hours to train a network to recognize a few milliseconds of speech.[7] If we could expand our system to the thousand-neuron level, it would be possible to learn simple speech recognition in real time.

Because the chip uses Ohm's law to multiply, charge conservation to add, device physics to create a threshold step, and a physical noise mechanism for random number generation, we can present training patterns to this chip about 100,000 times faster than the computer simulator. This factor, mostly due to the physical analog computation at this small network size, will increase with the size of the system due to its inherently parallel nature. It would also be possible to build fast special-purpose digital hardware to perform the multiply-accumulate calculations and do fast compares in parallel. Such hardware would take up considerably more silicon area but may be a good way to integrate neural network calculations into existing computer systems. If we could build a large VLSI learning system of, say, 10,000 neurons and 1,000,000 synapses, it would be about a billion times faster than a simulator on a 1 MIP machine. Presumably, such a system will be able to learn things beyond the capability of simulations even if they are run on supercomputers. However, there are several challenges to building these systems.

An algorithmic problem divorced from implementation is the effect of scaling to large size in highly connected networks. The learning time of such a system scales exponentially with the size of the problem.[8] The traditional way of handling complexity in large problems is to break them into smaller subpieces. An effective algorithm is yet to be discovered for doing learning in the modular, hierarchical networks which would be required to handle large problems.

Even from a technological viewpoint, modularity is necessary to manage the connectivity in a typical multiple chip system. A highly connected system, even if it could be built, would take too long to settle even considering the technology and parallel speedups available. Constraints such as power dissipation, capacitive loading across chips, and interchip communication are difficult to solve. If we succeed in these challenges, we will have the problem of presenting data to the system at extremely high rates amounting to several thousand (or more) bits every few microseconds. Biology solves these problems in the visual system, for example, by highly parallel communication via the optic nerve. It is unlikely that we will be able to use a million bit wide bus in our electronic system, however.

Can one take the weights learned by a learning system and simply load them onto a much simpler system with *programmable* rather than *adaptive* synapses? This is perhaps possible for smaller systems where analog inaccuracies and defects can be controlled. Modular networks provide a way of handling inaccuracies. However, for large analog systems, adaptation mechanisms are needed to maintain accuracy. Even if the accuracy were a few percent, a system of only a hundred neurons would be inaccurate across chips. In biological systems, if one were to place the connection strengths found in brain A onto the structures of brain B, the result would be chaos rather than a brain transplant. The robustness of neural systems depends on having the neurons and synapses adapt to the particular environment they find themselves in. Nevertheless, some amount of hard-wiring is probably possible in modular systems if it is modifiable by a trainable portion of the network. A speech recognition system may, for example, adapt in real time to the accents and timbre of a particular speaker. It is also likely that the system would require at least partial training beforehand for robustness.

We plan to design a larger version of our test chip containing both neurons and synapses which can form part of a still larger multiple chip network with the addition of chips containing only synapses. This next chip will have self-powered synapses so that each neuron need only signal its state rather than drive an unknown number of neurons from other chips. In addition, the noise generator will be improved so that true annealing is possible. We may also go further toward a fully analog chip[2] by having a variable gain neuron. Analog charge domain storage of weights and transport of states would further reduce the silicon area necessary but the technology required is not standard.

There are many challenges in scaling learning networks up to the $10^4$ neuron and $10^6$ synapse range although these large electronic learning networks will have on the order of a billionfold speed advantage over simulations based on serial computers. Thus they may be able to address many longstanding problems in artificial intelligence which have resisted attack by more conventional methods.

*References*

1. J. Alspector & R.B. Allen, "A neuromorphic VLSI learning system", in *Advanced Research in VLSI: Proceedings of the 1987 Stanford Conference.* edited by P. Losleben (MIT Press, Cambridge, MA, 1987) pp. 313-349.

2. J. Alspector, R.B. Allen, V. Hu, & S. Satyanarayana, "Stochastic learning networks and their electronic implementation", *Neural Information Processing Systems* (Denver, Nov. 1987) pp. 9-21.

3. D.H. Ackley, G.E. Hinton, & T.J. Sejnowski, "A learning algorithm for Boltzmann machines", *Cognitive Science* **9** (1985) pp. 147-169.

4. B. Widrow & M.E. Hoff, "Adaptive switching circuits", *IRE WESCON Convention Record* Part 4, (1960) pp. 96-104.

5. F. Rosenblatt, *Principles of neurodynamics: Perceptrons and the theory of brain mechanisms.* Spartan Books, Washington, D.C. (1961).

6. J. Alspector, "A VLSI approach to neural-style information processing", in *VLSI Signal Processing III.* edited by R.W. Brodersen and H.S. Moscovitz (IEEE Press, New York, 1988) pp. 232-243.

7. T.K. Landauer, C. Kamm, & S. Singhal, "Teaching a minimally structured back-propagation network to recognize speech sounds", *Proceedings of the Cognitive Science Society* (Seattle, Aug. 1987) pp. 531-536.

8. G. Tesauro & B. Janssens, "Scaling relationships in back-propagation learning", *Complex Systems* **2** (1988) pp. 39-44.

## Footnotes

* Permanent address: University of California, Berkeley; EE Dep't; Cory Hall; Berkeley, CA 94720
